# ENCODING GEOMETRIC INVARIANCES IN HIGHER-ORDER NEURAL NETWORKS

C.L. Giles
Air Force Office of Scientific Research, Bolling AFB, DC 20332

R.D. Griffin
Naval Research Laboratory, Washington, DC 20375-5000

T. Maxwell
Sachs-Freeman Associates, Landover, MD 20785

## ABSTRACT

We describe a method of constructing higher-order neural networks that respond invariantly under geometric transformations on the input space. By requiring each unit to satisfy a set of constraints on the interconnection weights, a particular structure is imposed on the network. A network built using such an architecture maintains its invariant performance independent of the values the weights assume, of the learning rules used, and of the form of the nonlinearities in the network. The invariance exhibited by a first-order network is usually of a trivial sort, e.g., responding only to the average input in the case of translation invariance, whereas higher-order networks can perform useful functions and still exhibit the invariance. We derive the weight constraints for translation, rotation, scale, and several combinations of these transformations, and report results of simulation studies.

## INTRODUCTION

A persistent difficulty for pattern recognition systems is the requirement that patterns or objects be recognized independent of irrelevant parameters or distortions such as orientation (position, rotation, aspect), scale or size, background or context, doppler shift, time of occurrence, or signal duration. The remarkable performance of humans and other animals on this problem in the visual and auditory realms is often taken for granted, until one tries to build a machine with similar performance. Though many methods have been developed for dealing with these problems,[1] we have classified them into two categories: 1) preprocessing or transformation (inherent) approaches, and 2) case-specific or "brute force" (learned) approaches. Common transformation techniques include: Fourier, Hough, and related transforms; moments; and Fourier descriptors of the input signal. In these approaches the signal is usually transformed so that the subsequent processing ignores arbitrary parameters such as scale, translation, etc. In addition, these techniques are usually computationally expensive and are sensitive to noise in the input signal. The "brute force" approach is exemplified by training a device, such as a perceptron, to classify a pattern independent of it's position by presenting the

training pattern at all possible positions. MADALINE machines[2] have been shown to perform well using such techniques. Often, this type of invariance is pattern specific, does not easily generalize to other patterns, and depends on the type of learning algorithm employed. Furthermore, a great deal of time and energy is spent on learning the invariance, rather than on learning the signal. We describe a method that has the advantage of inherent invariance but uses a higher-order neural network approach that must learn only the desired signal. Higher-order units have been shown to have unique computational strengths and are quite amenable to the encoding of a priori knowledge.[3-7]

## MATHEMATICAL DEVELOPMENT

Our approach is similar to the group invariance approach,[8,10] although we make no appeal to group theory to obtain our results. We begin by selecting a transformation on the input space, then require the output of the unit to be invariant to the transformation. The resulting equations yield constraints on the interconnection weights, and thus imply a particular form or structure for the network architecture.

For the i-th unit $y_i$ of order M defined on a discrete input space, let the output be given by

$$y_i[W_i^M(x), p(x)] = f( w_i^0 + \Sigma\, w_i^1(x_1)\, p(x_1)$$

$$+ \Sigma\Sigma\, w_i^2(x_1,x_2)\, p(x_1)\, p(x_2) + \ldots$$

$$+ \Sigma\ldots\Sigma\, w_i^M(x_1,\ldots x_M)\, p(x_1)..p(x_M)\, ), \qquad (1)$$

where $p(x)$ is the input pattern or signal function (sometimes called a pixel) evaluated at position vector $x$, $w_i^m(x_1,\ldots x_m)$ is the weight of order m connecting the outputs of units at $x_1$, $x_2$,..$x_m$ to the i-th unit, i.e., it correlates m values, $f(u)$ is some threshold or sigmoid output function, and the summations extend over the input space. $W_i^M(x)$ represents the entire set of weights associated with the i-th unit. These units are equivalent to the sigma-pi units[a] defined by Rumelhart, Hinton, and Williams.[7] Systems built from these units suffer from a combinatorial explosion of terms, hence are more complicated to build and train. To reduce the severity of this problem, one can limit the range of the interconnection weights or the number of orders, or impose various other constraints. We find that, in addition to the advantages of inherent invariance, imposing an invariance constraint on Eq. (1) reduces the number of allowed

---

[a]The sigma-pi neural networks are multi-layer networks with higher-order terms in any layer. As such, most of the neural networks described here can be considered as a special case of the sigma-pi units. However, the sigma-pi units as originally formulated did not have invariant weight terms, though it is quite simple to incorporate such invariances in these units.

weights, thus simplifying the architecture and shortening the training time.

We now define what we mean by invariance. The output of a unit is invariant with respect to the transformation $T$ on the input pattern if[9]

$$T[y_i(W_i{}^M, p(x))] = y(W_i{}^M, T[p(x)]) = y(W_i{}^M, p(x)) \qquad (2)$$

An example of the class of invariant response defined by Eq. (2) would be invariant detection of an object in the receptive field of a panning or zooming camera. An example of a different class would be invariant detection of an object that is moving within the field of a fixed camera. One can think of this latter case as consisting of a fixed field of "noise" plus a moving field that contains only the object of interest. If the detection system does not respond to the fixed field, then this latter case is included in Eq. (2).

To illustrate our method we derive the weight constraints for one-dimensional translation invariance. We will first switch to a continuous formulation, however, for reasons of simplicity and generality, and because it is easier to grasp the physical significance of the results, although any numerical simulation requires a discrete formulation and has significant implications for the implementation of our results. Instead of an index i, we now keep track of our units with the continuous variable u. With these changes Eq. (2) now becomes

$$y[u; W^M(x), p(x)] = f( w^0 + \int dx_1 \, w^1(u; x_1) \, p(x_1) + \dots$$

$$+ \int \dots \int dx_1 \dots dx_M \, w^M(u; x_1, \dots x_M) \, p(x_1) \dots p(x_M) ), \qquad (3)$$

The limits on the integrals are defined by the problem and are crucial in what follows. Let $T$ be a translation of the input pattern by $-x_0$, so that

$$T[p(x)] = p(x+x_0) \qquad (4)$$

where $x_0$ is the translation of the input pattern. Then, from eq (2),

$$Ty[u; W^M(x), p(x)] = y[u; W^M(x), p(x+x_0)] = y[u; W^M(x), p(x)] \qquad (5)$$

Since $p(x)$ is arbitrary we must impose term-by-term equality in the argument of the threshold function; i.e.,

$$\int dx_1 \, w^1(u; x_1) \, p(x_1) = \int dx_1 \, w^1(u; x_1) \, p(x_1+x_0), \qquad (5a)$$

$$\int \int dx_1 \, dx_2 \, w^2(u; x_1, x_2) \, p(x_1) \, p(x_2) =$$

$$\int \int dx_1 \, dx_2 \, w^2(u; x_1, x_2) \, p(x_1+x_0) \, p(x_2+x_0), \qquad (5b)$$

etc.

Making the substitutions $x_1 \rightarrow x_1\text{-}x_0$, $x_2 \rightarrow x_2\text{-}x_0$, etc, we find that

$$\int dx_1 \; w^1(u;x_1) \; p(x_1) = \int dx_1 \; w^1(u;x_1\text{-}x_0) \; p(x_1), \qquad (6a)$$

$$\int\int dx_1 \; dx_2 \; w^2(u;x_1,x_2) \; p(x_1) \; p(x_2) =$$

$$\int\int dx_1 \; dx_2 \; w^2(u;x_1\text{-}x_0,x_2\text{-}x_0) \; p(x_1) \; p(x_2), \qquad (6b)$$

etc.

Note that the limits of the integrals on the right hand side must be adjusted to satisfy the change-of-variables. If the limits on the integrals are infinite or if one imposes some sort of periodic boundary condition, the limits of the integrals on both sides of the equation can be set equal. We will assume in the remainder of this paper that these conditions can be met; normally this means the limits of the integrals extend to infinity. (In an implementation, it is usually impractical or even impossible to satisfy these requirements, but our simulation results indicate that these networks perform satisfactorily even though the regions of integration are not identical. This question must be addressed for each class of transformation; it is an integral part of the implementation design.) Since the functions $p(x)$ are arbitrary and the regions of integration are the same, the weight functions must be equal. This imposes a constraint on the functional form of the weight functions or, in the discrete implementation, limits the allowed connections and thus the number of weights. In the case of translation invariance, the constraint on the functional form of the weight functions requires that

$$w^1(u;x_1) = w^1(u;x_1\text{-}x_0), \qquad (7a)$$

$$w^2(u;x_1,x_2) = w^2(u;x_1\text{-}x_0,x_2\text{-}x_0), \qquad (7b)$$

etc.

These equations imply that the first order weight is independent of input position, and depends only on the output position $u$. The second order weight is a function only of vector differences,[10] i.e.,

$$w^1(u;x_1) = w^1(u), \qquad (8a)$$

$$w^2(u;x_1,x_2) = w^2(u;x_1\text{-}x_2). \qquad (8b)$$

For a discrete implementation with N input units (pixels) fully connected to an output unit, this requirement reduces the number of second-order weights from order $N^2$ to order N, i.e., only weights for differences of indexes are needed rather than all unique pair combinations. Of course, this advantage is multiplied as the number of fully-connected output units increases.

## FURTHER EXAMPLES

We have applied these techniques to several other transformations of interest. For the case of transformation of scale

define the scale operator S such that

$$Sp(\mathbf{x}) = a^n p(a\mathbf{x}) \qquad (9)$$

where a is the scale factor, and **x** is a vector of dimension n. The factor $a^n$ is used for normalization purposes, so that a given figure always contains the same "energy" regardless of its scale. Application of the same procedure to this transformation leads to the following constraints on the weights:

$$w^1(u;x_1/a) = w^1(u;x_1), \qquad (10a)$$

$$w^2(u;x_1/a,x_2/a) = w^2(u;x_1,x_2), \qquad (10b)$$

$$w^3(u;x_1/a,x_2/a,x_3/a) = w^3(u;x_1,x_2,x_3), \text{ etc.} \qquad (10c)$$

Consider a two-dimensional problem viewed in polar coordinates (r,t). A set of solutions to these constraints is

$$w^1(u;r_1,t_1) = w^1(u;t_1), \qquad (11a)$$

$$w^2(u;r_1,r_2;t_1,t_2) = w^2(u;r_1/r_2;t_1,t_2), \qquad (11b)$$

$$w^3(u;r_1,r_2,r_3;t_1,t_2,t_3) = w^3(u;(r_1-r_2)/r_3;t_1,t_2,t_3). \qquad (11c)$$

Note that with increasing order comes increasing freedom in the selection of the functional form of the weights. Any solution that satisfies the constraint may be used. This gives the designer additional freedom to limit the connection complexity, or to encode special behavior into the net architecture. An example of this is given later when we discuss combining translation and scale invariance in the same network.

Now consider a change of scale for a two-dimensional system in rectangular coordinates, and consider only the second-order weights. A set of solutions to the weight constraint is:

$$w^2(u;x_1,y_1;x_2,y_2) = w^2(u;x_1/y_1;x_2/y_2), \qquad (12a)$$

$$w^2(u;x_1,y_1;x_2,y_2) = w^2(u;x_1/x_2;y_1/y_2), \qquad (12b)$$

$$w^2(u;x_1,y_1;x_2,y_2) = w^2(u;(x_1-x_2)/(y_1-y_2)), \text{ etc.} \qquad (12c)$$

We have done a simulation using the form of Eq. (12b). The simulation was done using a small input space (8x8) and one output unit. A simple least-mean-square (back-propagation) algorithm was used for training the network. When taught to distinguish the letters T and C at one scale, it distinguished them at changes of scale of up to 4X with about 15 percent maximum degradation in the output strength. These results are quite encouraging because no special effort was required to make the system work, and no corrections or modifications were made to account for the boundary condition requirements as discussed near Eq. (6). This and other simulations are discussed further later.

As a third example of a geometric transformation, consider the case of rotation about the origin for a two-dimensional space in polar coordinates. One can readily show that the weight constraints

are satisfied if

$$w^1(u;r_1,t_1) = w^1(u;r_1),\tag{13a}$$

$$w^2(u;r_1,r_2;t_1,t_2) = w^2(u;r_1,r_2;t_1-t_2),\text{ etc.}\tag{13b}$$

These results are reminiscent of the results for translation invariance. This is not uncommon: seemingly different problems often have similar constraint requirements if the proper change of variable is made. This can be used to advantage when implementing such networks but we will not discuss it further here.

An interesting case arises when one considers combinations of invariances, e.g., scale and translation. This raises the question of the effect of the order of the transformations, i.e., is scale followed by translation equivalent to translation followed by scale? The obvious answer is no, yet for certain cases the order is unimportant. Consider first the case of change-of-scale by a, followed by a translation $x_0$; the constraints on the weights up to second order are:

$$w^1(u;x_1) = w^1(u;(x_1-x_0)/a),\tag{14a}$$

$$w^2(u;x_1,x_2) = w^2(u;(x_1-x_0)/a,(x_2-x_0)/a),\tag{14b}$$

and for translation followed by scale the constraints are:

$$w^1(u;x_1) = w^1(u;(x_1/a)-x_0),\text{ and}\tag{15a}$$

$$w^2(u;x_1,x_2) = w^2(u;(x_1/a)-x_0,(x_2/a)-x_0).\tag{15b}$$

Consider only the second-order weights for the two-dimensional case. Choose rectangular coordinate variables $(x,y)$ so that the translation is given by $(x_0,y_0)$. Then

$$w^2(u;x_1,y_1;x_2,y_2) =$$
$$w^2(u;(x_1/a)-x_0,(y_1/a)-y_0;(x_2/a)-x_0,(y_2/a)-y_0),\tag{16a}$$

or

$$w^2(u;x_1,y_1;x_2,y_2) =$$
$$w^2(u;(x_1-x_0)/a,(y_1-y_0)/a;(x_2-x_0)/a,(y_2-y_0)/a).\tag{16b}$$

If we take as our solution

$$w^2(u;x_1,y_1;x_2,y_2) = w^2(u;(x_1-x_2)/(y_1-y_2)),\tag{17}$$

then $w^2$ is invariant to scale and translation, and the order is unimportant. With higher-order weights one can be even more adventurous.

As a final example consider the case of a change of scale by a factor a and rotation about the origin by an amount $t_0$ for a two-dimensional system in polar coordinates. (Note that the order of transformation makes no difference.) The weight constraints up to second order are:

$$w^1(u;r_1,t_1) = w^1(u;r_1/a,t_1-t_0),\text{ and}\tag{18a}$$

$$w^2(r_1,t_1;r_2,t_2) = w^2(u;r_1/a,t_1-t_0;r_2/a,t_2-t_0).\qquad(18b)$$

The first-order constraint requires that $w^1$ be independent of the
input variables, but for the second-order term one can obtain a more
useful solution:

$$w^2(u;r_1,t_1;r_2,t_2) = w^2(u;r_1/r_2;t_1-t_2).\qquad(19)$$

This implies that with second-order weights, one can construct a unit
that is insensitive to changes in scale and rotation of the input
space.  How useful it is depends upon the application.

## SIMULATION RESULTS

We have constructed several higher-order neural networks that
demonstrated invariant response to transformations of scale and of
translation of the input patterns.  The systems were small,
consisting of less than 100 input units, were constructed from
second-and first-order units, and contained only one, two, or three
layers.  We used a back-propagation algorithm modified for the
higher-order (sigma-pi) units.  The simulation studies are still in
the early stages, so the performance of the networks has not been
thoroughly investigated.  It seems safe to say, however, that there
is much to be gained by a thorough study of these systems.  For
example, we have demonstrated that a small system of second-order
units trained to distinguish the letters T and C at one scale can
continue to distinguish them over changes in scale of factors of at
least four without retraining and with satisfactory performance.
Similar performance has been obtained for the case of translation
invariance.

Even at this stage, some interesting facets of this approach are
becoming clear: 1) Even with the constraints imposed by the
invariance, it is usually necessary to limit the range of connections
in order to restrict the complexity of the network.  This is often
cited as a problem with higher-order networks, but we take the view
that one can learn a great deal more about the nature of a problem by
examining it at this level rather than by simply training a network
that has a general-purpose architecture.  2)  The higher-order
networks seem to solve problems in an elegant and simple manner.
However, unless one is careful in the design of the network, it
performs worse than a simpler conventional network when there is
noise in the input field.  3) Learning is often "quicker" than in a
conventional approach, although this is highly dependent on the
specific problem and implementation design.  It seems that a tradeoff
can be made:  either faster learning but less noise robustness, or
slower learning with more robust performance.

## DISCUSSION

We have shown a simple way to encode geometric invariances into
neural networks (instead of training them), though to be useful the
networks must be constructed of higher-order units.  The invariant
encoding is achieved by restricting the allowable network

architectures and is independent of learning rules and the form of the sigmoid or threshold functions. The invariance encoding is normally for an entire layer, although it can be on an individual unit basis. It is easy to build one or more invariant layers into a multi-layer net, and different layers can satisfy different invariance requirements. This is useful for operating on internal features or representations in an invariant manner. For learning in such a net, a multi-layered learning rule such as generalized back-propagation[7] must be used. In our simulations we have used a generalized back-propagation learning rule to train a two-layer system consisting of a second-order, translation-invariant input layer and a first-order output layer. Note that we have not shown that one can not encode invariances into layered first-order networks, but the analysis in this paper implies that such invariance would be dependent on the form of the sigmoid function.

When invariances are encoded into higher-order neural networks, the number of interconnections required is usually reduced by orders of powers of N where N is the size of the input. For example, a fully connected, first-order, single-layer net with a single output unit would have order N interconnections; a similar second-order net, order $N^2$. If this second-order net (or layer) is made shift invariant, the order is reduced to N. The number of multiplies and adds is still of order $N^2$.

We have limited our discussion in this paper to geometric invariances, but there seems to be no reason why temporal or other invariances could not be encoded in a similar manner.

## REFERENCES

1. D.H. Ballard and C.M. Brown, Computer Vision (Prentice-Hall, Englewood Cliffs, NJ, 1982).

2. B. Widrow, IEEE First Intl. Conf. on Neural Networks, 87TH0191-7, Vol. 1, p. 143, San Diego, CA, June 1987.

3. J.A. Feldman, Biological Cybernetics 46, 27 (1982).

4. C.L. Giles and T. Maxwell, Appl. Optics 26, 4972 (1987).

5. G.E. Hinton, Proc. 7th Intl. Joint Conf. on Artificial Intelligence, ed. A. Drina, 683 (1981).

6. Y.C. Lee, G. Doolen, H.H. Chen, G.Z. Sun, T. Maxwell, H.Y. Lee, C.L. Giles, Physica 22D, 276 (1986).

7. D.E. Rumelhart, G.E. Hinton, and R.J. Williams, Parallel Distributed Processing, Vol. 1, Ch. 8, D.E. Rumelhart and J.L. McClelland, eds., (MIT Press, Cambridge, 1986).

8.  T. Maxwell, C.L. Giles, Y.C. Lee, and H.H. Chen, Proc. IEEE
    Intl. Conf. on Systems, Man, and Cybernetics, 86CH2364-8, p.
    627, Atlanta, GA,  October 1986.

9.  W. Pitts and W.S. McCulloch, Bull. Math. Biophys. 9, 127
    (1947).

10. M. Minsky and S. Papert, Perceptrons (MIT Press, Cambridge,
    Mass., 1969).
